# Theory of matching pursuit

**Zakria Hussain and John Shawe-Taylor**
Department of Computer Science
University College London, UK
{z.hussain,j.shawe-taylor}@cs.ucl.ac.uk

## Abstract

We analyse matching pursuit for kernel principal components analysis (KPCA) by proving that the sparse subspace it produces is a sample compression scheme. We show that this bound is tighter than the KPCA bound of Shawe-Taylor et al [7] and highly predictive of the size of the subspace needed to capture most of the variance in the data. We analyse a second matching pursuit algorithm called kernel matching pursuit (KMP) which does not correspond to a sample compression scheme. However, we give a novel bound that views the choice of subspace of the KMP algorithm as a compression scheme and hence provide a VC bound to upper bound its future loss. Finally we describe how the same bound can be applied to other matching pursuit related algorithms.

## 1 Introduction

Matching pursuit refers to a family of algorithms that generate a set of bases for learning in a greedy fashion. A good example of this approach is the matching pursuit algorithm [4]. Viewed from this angle sparse kernel principal components analysis (PCA) looks for a small number of kernel basis vectors in order to maximise the Rayleigh quotient. The algorithm was proposed by [8][1] and motivated by matching pursuit [4], but to our knowledge sparse PCA has not been analysed theoretically. In this paper we show that sparse PCA (KPCA) is a sample compression scheme and can be bounded using the size of the compression set [3, 2] which is the set of training examples used in the construction of the KPCA subspace. We also derive a more general framework for this algorithm that uses the principle "maximise Rayleigh quotient and deflate". A related algorithm called kernel matching pursuit (KMP) [10] is a sparse version of least squares regression but without the property of being a compression scheme. However, we use the number of basis vectors constructed by KMP to help upper bound the loss of the KMP algorithm using the VC dimension. This bound is novel in that it is applied in an empirically chosen low dimensional hypothesis space and applies independently of the actual dimension of the ambient feature space (including one constructed from the Gaussian kernel). In both cases we illustrate the use of the bounds on real and/or simulated data.

Finally we also show that the KMP bound can be applied to a sparse kernel canonical correlation analysis that uses a similar matching pursuit technique. We do not describe the algorithm here due to space constraints and only concentrate on theoretical results. We begin with preliminary definitions.

## 2 Preliminary definitions

Throughout the paper we consider learning from samples of data. For the regression section the data is a sample $S = \{(\mathbf{x}_i, y_i)\}_{i=1}^{m}$ of input-output pairs drawn from a joint space $\mathcal{X} \times \mathcal{Y}$ where $\mathbf{x} \in \mathbb{R}^n$ and $y \in \mathbb{R}$. For the principal components analysis the data is a sample $S = \{\mathbf{x}_i\}_{i=1}^{m}$ of

multivariate examples drawn from a space $\mathcal{X}$. For simplicity we always assume that the examples are already projected into the kernel defined feature space, so that the kernel matrix $\mathbf{K}$ has entries $\mathbf{K}[i,j] = \langle \mathbf{x}_i, \mathbf{x}_j \rangle$. The notation $\mathbf{K}[i,:]$ and $\mathbf{K}[:,i]$ will denote the $i$th row and $i$th column of the matrix $\mathbf{K}$, respectively. When using a set of indices $\mathbf{i} = \{i_1, \ldots, i_k\}$ (say) then $\mathbf{K}[\mathbf{i},\mathbf{i}]$ denotes the square matrix defined solely by the index set $\mathbf{i}$. The transpose of a matrix $\mathbf{X}$ or vector $\mathbf{x}$ is denoted by $\mathbf{X}'$ or $\mathbf{x}'$ respectively. The input data matrix $\mathbf{X}$ will contain examples as row vectors.

For analysis purposes we assume that the training examples are generated i.i.d. according to an unknown but fixed probability distribution that also governs the generation of the test data. Expectation over the training examples (empirical average) is denoted by $\hat{\mathcal{E}}[\cdot]$, while expectation with respect to the underlying distribution is denoted $\mathcal{E}[\cdot]$. For the sample compression analysis the **compression function** $\Lambda$ induced by a sample compression learning algorithm $A$ on training set $S$ is the map $\Lambda : S \longmapsto \Lambda(S)$ such that the *compression set* $\Lambda(S) \subset S$ is returned by $A$. A **reconstruction function** $\Phi$ is a mapping from a compression set $\Lambda(S)$ to a set $F$ of functions $\Phi : \Lambda(S) \longmapsto F$. Let $A(S)$ be the function output by learning algorithm $A$ on training set $S$. Therefore, a sample compression scheme is a reconstruction function $\Phi$ mapping a compression set $\Lambda(S)$ to some set of functions $F$ such that $A(S) = \Phi(\Lambda(S))$. If $F$ is the set of Boolean-valued or Real-valued functions then the sample compression scheme is said to be a classification or regression algorithm, respectively.

## 2.1 Sparse kernel principal components analysis

Principal components analysis [6] can be expressed as the following maximisation problem:

$$\max_{\mathbf{w}} \frac{\mathbf{w}'\mathbf{X}'\mathbf{X}\mathbf{w}}{\mathbf{w}'\mathbf{w}}, \tag{1}$$

where $\mathbf{w}$ is the weight vector. In a sparse KPCA algorithm we would like to find a sparsely represented vector $\mathbf{w} = \mathbf{X}[\mathbf{i},:]'\tilde{\boldsymbol{\alpha}}$, that is a linear combination of a small number of training examples indexed by vector $\mathbf{i}$. Therefore making this substitution into Equation (1) we have the following sparse dual PCA maximisation problem,

$$\max_{\tilde{\boldsymbol{\alpha}}} \frac{\tilde{\boldsymbol{\alpha}}'\mathbf{X}[\mathbf{i},:]\mathbf{X}'\mathbf{X}\mathbf{X}[\mathbf{i},:]'\tilde{\boldsymbol{\alpha}}}{\tilde{\boldsymbol{\alpha}}'\mathbf{X}[\mathbf{i},:]\mathbf{X}[\mathbf{i},:]'\tilde{\boldsymbol{\alpha}}},$$

which is equivalent to sparse kernel PCA (SKPCA) with sparse kernel matrix $\mathbf{K}[:,\mathbf{i}]' = \mathbf{X}[\mathbf{i},:]\mathbf{X}'$,

$$\max_{\tilde{\boldsymbol{\alpha}}} \frac{\tilde{\boldsymbol{\alpha}}'\mathbf{K}[:,\mathbf{i}]'\mathbf{K}[:,\mathbf{i}]\tilde{\boldsymbol{\alpha}}}{\tilde{\boldsymbol{\alpha}}'\mathbf{K}[\mathbf{i},\mathbf{i}]\tilde{\boldsymbol{\alpha}}},$$

where $\tilde{\boldsymbol{\alpha}}$ is a sparse vector of length $k = |\mathbf{i}|$. Clearly maximising the quantity above will lead to the maximisation of the generalised eigenvalues corresponding to $\tilde{\boldsymbol{\alpha}}$ – and hence a sparse subset of the original KPCA problem. We would like to find the optimal set of indices $\mathbf{i}$. We proceed in a greedy manner (matching pursuit) in much the same way as [8]. The procedure involves choosing basis vectors that maximise the Rayleigh quotient without the set of eigenvectors. Choosing basis vectors iteratively until some pre-specified number of $k$ vectors are chosen. An orthogonalisation of the kernel matrix at each step ensures future potential basis vectors will be orthogonal to those already chosen. The quotient to maximise is:

$$\max \rho_i \quad = \quad \frac{\mathbf{e}_i'\mathbf{K}^2\mathbf{e}_i}{\mathbf{e}_i'\mathbf{K}\mathbf{e}_i}, \tag{2}$$

where $\mathbf{e}_i$ is the $i$th unit vector. After this maximisation we need to orthogonalise (deflate) the kernel matrix to create a projection into the space orthogonal to the basis vectors chosen to ensure we find the maximum variance of the data in the projected space. The deflation step can be carried out as follows. Let $\boldsymbol{\tau} = \mathbf{K}[:,i] = \mathbf{X}\mathbf{X}'\mathbf{e}_i$ where $\mathbf{e}_i$ is the $i$th unit vector. We know that primal PCA deflation can be carried out with respect to the features in the following way:

$$\hat{\mathbf{X}}' = \left(\mathbf{I} - \frac{\mathbf{u}\mathbf{u}'}{\mathbf{u}'\mathbf{u}}\right)\mathbf{X}',$$

where $\mathbf{u}$ is the projection directions defined by the chosen eigenvector and $\hat{\mathbf{X}}$ is the deflated matrix. However, in sparse KPCA, $\mathbf{u} = \mathbf{X}'\mathbf{e}_i$ because the projection directions are simply the examples in $\mathbf{X}$. Therefore, for sparse KPCA we have:

$$\hat{\mathbf{X}}\hat{\mathbf{X}}' = \mathbf{X}\left(\mathbf{I} - \frac{\mathbf{u}\mathbf{u}'}{\mathbf{u}'\mathbf{u}}\right)\left(\mathbf{I} - \frac{\mathbf{u}\mathbf{u}'}{\mathbf{u}'\mathbf{u}}\right)\mathbf{X}' = \mathbf{X}\mathbf{X}' - \frac{\mathbf{X}\mathbf{X}'\mathbf{e}_i\mathbf{e}_i'\mathbf{X}\mathbf{X}'}{\mathbf{e}_i'\mathbf{X}\mathbf{X}'\mathbf{e}_i} = \mathbf{K} - \frac{\mathbf{K}[:,i]\mathbf{K}[:,i]'}{\mathbf{K}[i,i]}.$$

Therefore, given a kernel matrix $\mathbf{K}$ the deflated kernel matrix $\hat{\mathbf{K}}$ can be computed as follows:

$$\hat{\mathbf{K}} = \mathbf{K} - \frac{\boldsymbol{\tau}\boldsymbol{\tau}'}{\mathbf{K}[\mathbf{i}_k, \mathbf{i}_k]} \tag{3}$$

where $\boldsymbol{\tau} = \mathbf{K}[:, \mathbf{i}_k]$ and $\mathbf{i}_k$ denotes the latest element in the vector $\mathbf{i}$. The algorithm is presented below in Algorithm 1 and we use the notation $\mathbf{K}^{\cdot 2}$ to denote component wise squaring. Also, division of vectors are assumed to be component wise.

---

**Algorithm 1**: A matching pursuit algorithm for kernel principal components analysis (*i.e.*, sparse KPCA)

---

**Input:** Kernel $\mathbf{K}$, sparsity parameter $k > 0$.
 1: initialise $\mathbf{i} = [\,]$
 2: **for** $j = 1$ to $k$ **do**
 3:    Set $\mathbf{i}_j$ to index of $\max \left\{ \frac{(\mathbf{K}^{\cdot 2})'\mathbf{1}}{diag\{\mathbf{K}\}} \right\}$
 4:    set $\boldsymbol{\tau} = \mathbf{K}[:, \mathbf{i}_j]$ to deflate kernel matrix like so: $\mathbf{K} = \mathbf{K} - \frac{\boldsymbol{\tau}\boldsymbol{\tau}'}{\mathbf{K}[\mathbf{i}_j, \mathbf{i}_j]}$
 5: **end for**
 6: Compute $\tilde{\mathbf{K}}$ using $\mathbf{i}$ and Equation (5)
**Output:** Output sparse matrix approximation $\tilde{\mathbf{K}}$

---

This algorithm is presented in Algorithm 1 and is equivalent to the algorithm proposed by [8]. However, their motivation comes from the stance of finding a low rank matrix approximation of the kernel matrix. They proceed by looking for an approximation $\tilde{\mathbf{K}} = \mathbf{K}[:, \mathbf{i}]T$ for a set $\mathbf{i}$ such that the Frobenius norm between the trace residuals $\mathrm{tr}\{\mathbf{K} - \mathbf{K}[:, \mathbf{i}]T\} = \mathrm{tr}\{\mathbf{K} - \hat{\mathbf{K}}\}$ is minimal. Their algorithm finds the set of indices $\mathbf{i}$ and the projection matrix $T$. However, the use of $T$ in computing the low rank matrix approximation seems to imply the need for additional information from outside of the chosen basis vectors in order to construct this approximation. However, we show that a projection into the space defined solely by the chosen indices is enough to reconstruct the kernel matrix and does not require any extra information.[2] The projection is the well known Nyström method [11].

An orthogonal projection $P_{\mathbf{i}}(\phi(\mathbf{x}_j))$ of a feature vector $\phi(\mathbf{x}_j)$ into a subspace defined only by the set of indices $\mathbf{i}$ can be expressed as: $P_{\mathbf{i}}(\mathbf{x}_j) = \tilde{\mathbf{X}}'(\tilde{\mathbf{X}}\tilde{\mathbf{X}}')^{-1}\tilde{\mathbf{X}}\phi(\mathbf{x}_j)$, where $\tilde{\mathbf{X}} = \mathbf{X}[\mathbf{i}, :]$ are the $\mathbf{i}$ training examples from data matrix $\mathbf{X}$. It follows that,

$$\begin{aligned} P_{\mathbf{i}}(\mathbf{x}_j)'P_{\mathbf{i}}(\mathbf{x}_j) &= \phi(\mathbf{x}_j)'\tilde{\mathbf{X}}'(\tilde{\mathbf{X}}\tilde{\mathbf{X}}')^{-1}\tilde{\mathbf{X}}\tilde{\mathbf{X}}'(\tilde{\mathbf{X}}\tilde{\mathbf{X}}')^{-1}\tilde{\mathbf{X}}\phi(\mathbf{x}_j) \\ &= \mathbf{K}[\mathbf{i}, j]\mathbf{K}[\mathbf{i}, \mathbf{i}]^{-1}\mathbf{K}[j, \mathbf{i}], \end{aligned} \tag{4}$$

with $\mathbf{K}[\mathbf{i}, j]$ denoting the kernel entries between the index set $\mathbf{i}$ and the feature vector $\phi(\mathbf{x}_j)$. Giving us the following projection into the space defined by $\mathbf{i}$:

$$\tilde{\mathbf{K}} = \mathbf{K}[:, \mathbf{i}]\mathbf{K}[\mathbf{i}, \mathbf{i}]^{-1}\mathbf{K}[:, \mathbf{i}]'. \tag{5}$$

**Claim 1.** *The sparse kernel principal components analysis algorithm is a compression scheme.*

*Proof.* We can reconstruct the projection from the set of chosen indices $\mathbf{i}$ using Equation (4). Hence, $\mathbf{i}$ forms a compression set.  □

We now prove that Smola and Schölkopf's low rank matrix approximation algorithm [8] (without sub-sampling)[3] is equivalent to sparse kernel principal components analysis presented in this paper (Algorithm 1).

**Theorem 1.** *Without sub-sampling, Algorithm 1 is equivalent to Algorithm 2 of [8].*

*Proof.* Let $\mathbf{K}$ be the kernel matrix and let $\mathbf{K}[:, i]$ be the $i$th column of the kernel matrix. Assume $\mathbf{X}$ is the input matrix containing rows of vectors that have already been mapped into a higher dimensional feature space using $\phi$ such that $\mathbf{X} = (\phi(\mathbf{x}_1), \ldots, \phi(\mathbf{x}_m))'$. Smola and Schölkopf [8] state in section 4.2 of their paper that their algorithm 2 finds a low rank approximation of the kernel matrix such that it minimises the Frobenius norm $\|\mathbf{X} - \tilde{\mathbf{X}}\|^2_{\text{Frob}} = \text{tr}\{\mathbf{K} - \tilde{\mathbf{K}}\}$ where $\tilde{\mathbf{X}}$ is the low rank approximation of $\mathbf{X}$. Therefore, we need to prove that Algorithm 1 also minimises this norm.

We would like to show that the maximum reduction in the Frobenius norm between the kernel $\mathbf{K}$ and its projection $\tilde{\mathbf{K}}$ is in actual fact the choice of basis vectors that maximise the Rayleigh quotient and deflate according to Equation (3). At each stage we deflate by,

$$\mathbf{K} \quad = \quad \mathbf{K} - \frac{\boldsymbol{\tau}\boldsymbol{\tau}'}{\mathbf{K}[\mathbf{i}_k, \mathbf{i}_k]}.$$

The trace $\text{tr}\{\mathbf{K}\} = \sum_{i=1}^{m} \mathbf{K}[i, i]$ is the sum of the diagonal elements of matrix $\mathbf{K}$. Therefore,

$$\text{tr}\{\mathbf{K}\} \quad = \quad \text{tr}\{\mathbf{K}\} - \frac{\text{tr}\{\boldsymbol{\tau}\boldsymbol{\tau}'\}}{\mathbf{K}[\mathbf{i}_k, \mathbf{i}_k]} = \text{tr}\{\mathbf{K}\} - \frac{\text{tr}\{\boldsymbol{\tau}'\boldsymbol{\tau}\}}{\mathbf{K}[\mathbf{i}_k, \mathbf{i}_k]} = \text{tr}\{\mathbf{K}\} - \frac{\mathbf{K}^2[\mathbf{i}_k, \mathbf{i}_k]}{\mathbf{K}[\mathbf{i}_k, \mathbf{i}_k]}.$$

The last term of the final equation corresponds exactly to the Rayleigh quotient of Equation (2). Therefore the maximisation of the Rayleigh quotient does indeed correspond to the maximum reduction in the Frobenius norm between the approximated matrix $\tilde{\mathbf{X}}$ and $\mathbf{X}$. $\qquad\square$

## 2.2 A generalisation error bound for sparse kernel principal components analysis

We use the sample compression framework of [3] to bound the generalisation error of the sparse KPCA algorithm. Note that kernel PCA bounds [7] do not use sample compression in order to bound the true error. As pointed out above, we use the simple fact that this algorithm can be viewed as a *compression scheme*. No side information is needed in this setting and a simple application of [3] is all that is required. That said the usual application of compression bounds has been for classification algorithms, while here we are considering a subspace method.

**Theorem 2.** *Let $\mathcal{A}_k$ be any learning algorithm having a reconstruction function that maps compression sets to subspaces. Let $m$ be the size of the training set $S$, let $k$ be the size of the compression set, let $\hat{\mathcal{E}}_{m-k}[\ell(\mathcal{A}_k(S))]$ be the residual loss between the $m - k$ points outside of the compression set and their projections into a subspace, then with probability $1 - \delta$, the expected loss $\mathcal{E}[\ell(\mathcal{A}_k(S))]$ of algorithm $\mathcal{A}_k$ given any training set $S$ can be bounded by,*

$$\mathcal{E}[\ell(\mathcal{A}_k(S))] \quad \leq \quad \min_{1 \leq t \leq k} \left[ \hat{\mathcal{E}}_{m-t}[\ell(\mathcal{A}_t(S))] + \sqrt{\frac{R}{2(m-t)} \left[ t \ln \left( \frac{em}{t} \right) + \ln \left( \frac{2m}{\delta} \right) \right]} \right],$$

*where $\ell(\cdot) \geq 0$ and $R = \sup \ell(\cdot)$.*

*Proof.* Consider the case where we have a compression set of size $k$. Then we have $\binom{m}{k}$ different ways of choosing the compression set. Given $\delta$ confidence we apply Hoeffding's bound to the $m - k$ points not in the compression set once for each choice by setting it equal to $\delta / \binom{m}{k}$. Solving for $\epsilon$ gives us the theorem when we further apply a factor $1/m$ to $\delta$ to ensure one application for each possible choice of $k$. The minimisation over $t$ chooses the best value making use of the fact that using more dimensions can only reduce the expected loss on test points. $\qquad\square$

We now consider the application of the above bound to sparse KPCA. Let the corresponding loss function be defined as

$$\ell(\mathcal{A}_t(S))(\mathbf{x}) \quad = \quad \|\mathbf{x} - P_{\mathbf{i}_t}(\mathbf{x})\|^2,$$

where $\mathbf{x}$ is a test point and $P_{\mathbf{i}_t}(\mathbf{x})$ its projection into the subspace determined by the set $\mathbf{i}_t$ of indices returned by $\mathcal{A}_t(S)$. Thus we can give a more specific loss bound in the case where we use a Gaussian kernel in the sparse kernel principal components analysis.

**Corollary 1** (Sample compression bound for sparse KPCA). *Using a Gaussian kernel and all of the definitions from Theorem 2, we get the following bound:*

$$\mathcal{E}[\ell(\mathcal{A}(S))] \quad \leq \quad \min_{1 \leq t \leq k} \left[ \frac{1}{m-t} \sum_{i=1}^{m-t} \|\mathbf{x}_i - P_{\mathbf{i}_t}(\mathbf{x}_i)\|^2 + \sqrt{\frac{1}{2(m-t)} \left[ t \ln \left( \frac{em}{t} \right) + \ln \left( \frac{2m}{\delta} \right) \right]} \right],$$

Note that $R$ corresponds to the smallest radius of a ball that encloses all of the training points. Hence, for the Gaussian kernel $R$ equals 1. We now compare the sample compression bound proposed above for sparse KPCA with the kernel PCA bound introduced by [7]. The left hand side of Figure 1 shows plots for the test error residuals (for the Boston housing data set) together with its upper bounds computed using the bound of [7] and the sample compression bound of Corollary 1. The sample compression bound is much tighter than the KPCA bound and also non-trivial (unlike the KPCA bound).

The sample compression bound is at its lowest point after 43 basis vectors have been added. We speculate that at this point the "true" dimensions of the data have been found and that all other dimensions correspond to "noise". This corresponds to the point at which the plot of residual becomes linear, suggesting dimensions with uniform noise. We carry out an extra toy experiment to help assess whether or not this is true and to show that the sample compression bound can help indicate when the principal components have captured most of the actual data. The right hand side plot of Figure 1 depicts the results of a toy experiment where we randomly sampled 1000 examples with 450 dimensions from a Gaussian distribution with zero mean and unit variance. We then ensured that 50 dimensions contained considerably larger eigenvalues than the remaining 400. From the right plot of Figure 1 we see that the test residual keeps dropping at a constant rate after 50 basis vectors have been added. The compression bound picks 46 dimensions with the largest eigenvalues, however, the KPCA bound of [7] is much more optimistic and is at its lowest point after 30 basis vectors, suggesting erroneously that SKPCA has captured most of the data in 30 dimensions. Therefore, as well as being tighter and non-trivial, the compression bound is much better at predicting the best choice for the number of dimensions to use with sparse KPCA. Note that we carried out this experiment without randomly permuting the projections into a subspace because SKPCA is rotation invariant and will always choose the principal components with the largest eigenvalues.

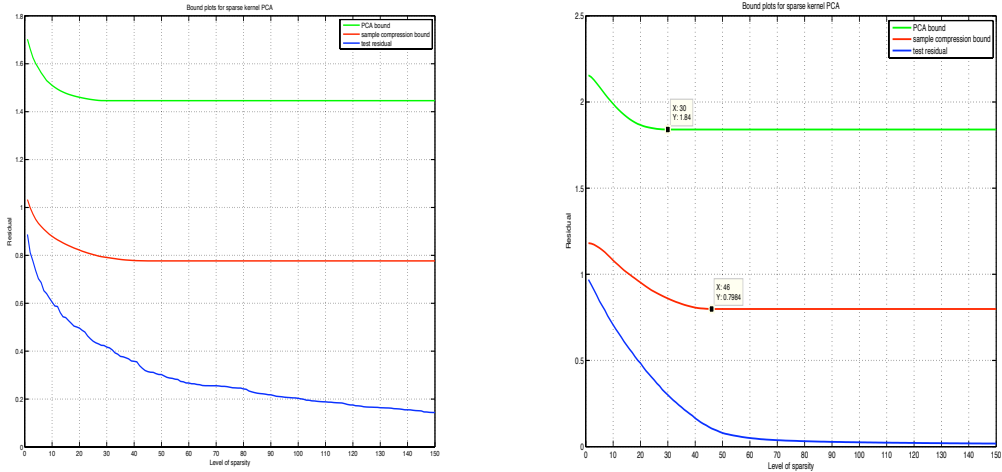

Figure 1: Bound plots for sparse kernel PCA comparing the sample compression bound proposed in this paper and the already existing PCA bound. The plot on the left hand side is for the Boston Housing data set and the plot on the right is for a Toy experiment with 1000 training examples (and 450 dimensions) drawn randomly from a Gaussian distribution with zero mean and unit variance.

## 3 Kernel matching pursuit

Unfortunately, the theory of the last section, where we gave a sample compression bound for SKPCA cannot be applied to KMP. This is because the algorithm needs information from outside of the compression set in order to construct its regressors and make predictions. However, we can use a VC argument together with a sample compression trick in order to derive a bound for KMP in terms of the level of sparsity achieved, by viewing the sparsity achieved in the feature space as a

compression scheme. Please note that we do not derive or reproduce the KMP algorithm here and advise the interested reader to read the manuscript of [10] for the algorithmic details.

## 3.1 A generalisation error bound for kernel matching pursuit

VC bounds have commonly been used to bound learning algorithms whose hypothesis spaces are infinite. One problem with these results is that the VC-dimension can sometimes be infinite even in cases where learning is successful (*e.g.*, the SVM). However, in this section we can avoid this issue by making use of the fact that the VC-dimension of the set of linear threshold functions is simply the dimensionality of the function class. In the kernel matching pursuit algorithm this translates directly into the number of basis vectors chosen and hence a standard VC argument.

The natural loss function for KMP is regression – however in order to use standard VC bounds we map the regression loss into a classification loss in the following way.

**Definition 1.** *Let $S \sim \mathcal{D}$ be a regression training sample generated iid from a fixed but unknown probability distribution $\mathcal{D}$. Given the error $\ell(f) = |f(\mathbf{x}) - y|$ for a regression function $f$ between training example $\mathbf{x}$ and regression output $y$ we can define, for some fixed positive scalar $\alpha \in \mathbb{R}$, the corresponding true classification loss (error) as*

$$\ell_\alpha(f) = \Pr_{(\mathbf{x},y)\sim\mathcal{D}} \{|f(\mathbf{x}) - y| > \alpha\}.$$

*Similarly, we can define the corresponding empirical classification loss as*

$$\hat{\ell}_\alpha(f) = \ell_\alpha^S(f) = \Pr_{(\mathbf{x},y)\sim S} \{|f(\mathbf{x}) - y| > \alpha\} = \mathbb{E}_{(\mathbf{x},y)\sim S} \{\mathbb{I}(|f(\mathbf{x}) - y| > \alpha)\},$$

*where $\mathbb{I}$ is the indicator function and $S$ is suppressed when clear from context.*

Now that we have a loss function that is binary we can make a simple sample compression argument, that counts the number of possible subspaces, together with a traditional VC style bound to upper bound the expected loss of KMP. To help keep the notation consistent with earlier definitions we will denote the indices of the chosen basis vectors by $\mathbf{i}$. The indices of $\mathbf{i}$ are chosen from the training sample $S$ and we denote $S_\mathbf{i}$ to be those samples indexed by the vector $\mathbf{i}$. Given these definitions and the bound of Vapnik and Chervonenkis [9] we can upper bound the true loss of KMP as follows.

**Theorem 3.** *Fix $\alpha \in \mathbb{R}$, $\alpha > 0$. Let $\mathcal{A}$ be the regression algorithm of KMP, $m$ the size of the training set $S$ and $k$ the size of the chosen basis vectors $\mathbf{i}$. Let $S$ be reordered so that the last $m - k$ points are outside of the set $\mathbf{i}$ and let $t = \sum_{i=m-k}^{m} \mathbb{I}(|f(\mathbf{x}_i) - y_i| > \alpha)$ be the number of errors for those points in $S \smallsetminus S_\mathbf{i}$. Then with probability $1 - \delta$ over the generation of the training set $S$ the expected loss $\mathcal{E}[\ell(\cdot)]$ of algorithm $\mathcal{A}$ can be bounded by,*

$$\mathcal{E}[\ell(\mathcal{A}(S))] \leq \frac{2}{m-k-t} \left[ (k+1)\log\left(\frac{4e(m-k-t)}{k+1}\right) + k\log\left(\frac{em}{k}\right) \right.$$
$$\left. + t\log\left(\frac{e(m-k)}{t}\right) + \log\left(\frac{2m^2}{\delta}\right) \right].$$

*Proof.* First consider a fixed size $k$ for the compression set and number of errors $t$. Let $S_1 = \{\mathbf{x}_{i_1}, \dots, \mathbf{x}_{i_k}\}$ be the set of $k$ training points chosen by the KMP regressor, $S_2 = \{\mathbf{x}_{i_{k+1}}, \dots, \mathbf{x}_{i_{k+t}}\}$ the set of points erred on in training and $\bar{S} = S \smallsetminus (S_1 \cup S_2)$ the points outside of the compression set ($S_1$) and training error set ($S_2$). Suppose that the first $k$ points form the compression set and the next $t$ are the errors of the KMP regressor. Since the remaining $m - k - t$ points $\bar{S}$ are drawn independently we can apply the VC bound [9] to the $\ell_\alpha$ loss to obtain the bound

$$\Pr\left\{\bar{S} : \ell_\alpha^{\bar{S}}(f) = 0, \ell_\alpha(f) > \epsilon\right\} \leq 2\left(\frac{4e(m-k-t)}{k+1}\right)^{k+1} 2^{-\epsilon(m-k-t)/2},$$

where we have made use of a bound on the number of dichotomies that can be generated by parallel hyperplanes [1], which is $\sum_{i=0}^{k+d-1} \binom{md-1}{i}$ which is $\leq \left(\frac{e(md-1)}{k+d-1}\right)^{k+d-1}$, where $d$ is the number of parallel hyperplanes and equals 2 in our case. We now need to consider all of the ways that the

$k$ basis vectors and $t$ error points might have occurred and apply the union bound over all of these possibilities. This gives the bound

$$\Pr\left\{S: \exists\, f \in \text{span}\{S_1\} \text{ s.t. } \ell_{\alpha}^{S_2}(f) = 1, \ell_{\alpha}^{\bar{S}}(f) = 0, \ell_{\alpha}(f) > \epsilon\right\}$$

$$\leq \binom{m}{k}\binom{m-k}{t} 2\left(\frac{4e(m-k-t)}{k+1}\right)^{k+1} 2^{-\epsilon(m-k-t)/2}. \tag{6}$$

Finally we need to consider all possible choices of the values of $k$ and $t$. The number of these possibilities is clearly upper bounded by $m^2$. Setting $m^2$ times the rhs of (6) equal to $\delta$ and solving for $\epsilon$ gives the result. $\qquad\square$

This is the first upper bound on the generalisation error for KMP that we are aware of and as such we cannot compare the bound against any others. Figure 2 plots the KMP test error against the loss

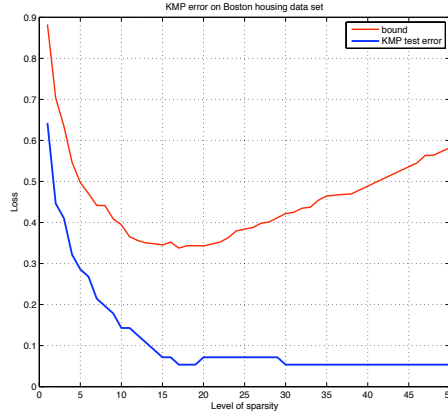

Figure 2: Plot of KMP bound against its test error. We used 450 examples for training and the 56 for testing. Bound was scaled down by a factor of 5.

bound given by Theorem 3. The bound value has been scaled by 5 in order to get the correct pictorial representation of the two plots. Figure 2 shows its minima directly coincides with the lowest test error (after 17 basis vectors). This motivates a training algorithm for KMP that would use the bound as the minimisation criteria and stop once the bound fails to become smaller. Hence, yielding a more automated training procedure.

## 4 Extensions

The same approach that we have used for bounding the performance of kernel matching pursuit can be used to bound a matching pursuit version of kernel canonical correlation analysis (KCCA) [6]. By choosing the basis vectors greedily to optimise the quotient:

$$\max_{i} \rho_i \;\; = \;\; \frac{\mathbf{e}_i' \mathbf{K}_x \mathbf{K}_y \mathbf{e}_i}{\sqrt{\mathbf{e}_i' \mathbf{K}_x^2 \mathbf{e}_i \mathbf{e}_i' \mathbf{K}_y^2 \mathbf{e}_i}},$$

and proceeding in the same manner as Algorithm 1 by deflating after each pair of basis vectors are chosen, we create a sparsely defined subspace within which we can run the standard CCA algorithm. This again means that the overall algorithm fails to be a compression scheme as side information is required. However, we can use the same approach described for KMP to bound the expected fit of the projections from the two views. The resulting bound has the following form.

**Theorem 4.** *Fix $\alpha \in \mathbb{R}$, $\alpha > 0$. Let $\mathcal{A}$ be the SKCCA algorithm, $m$ the size of the paired training sets $S^{\mathcal{X} \times \mathcal{Y}}$ and $k$ the cardinality of the set $\mathbf{i}$ of chosen basis vectors. Let $S^{\mathcal{X} \times \mathcal{Y}}$ be reordered so that the last $m - k$ paired data points are outside of the set $\mathbf{i}$ and define $t = \sum_{i=m-k}^{m} \mathbb{I}(|f_x(\mathbf{x}_i) - f_y(\mathbf{y}_i)| >$*

$\alpha$) *to be the number of errors for those points in $S^{\mathcal{X} \times \mathcal{Y}} \smallsetminus S_\mathbf{i}^{\mathcal{X} \times \mathcal{Y}}$, where $f_x$ is the projection function of the $\mathcal{X}$ view and $f_y$ the projection function of the $\mathcal{Y}$ view. Then with probability $1 - \delta$ over the generation of the paired training sets $S^{\mathcal{X} \times \mathcal{Y}}$ the expected loss $\mathcal{E}[\ell(\cdot)]$ of algorithm $\mathcal{A}$ can be bounded by,*

$$\mathcal{E}[\ell(\mathcal{A}(S))] \leq \frac{2}{m-k-t} \left[ (k+1)\log\left(\frac{4e(m-k-t)}{k+1}\right) + k\log\left(\frac{em}{k}\right) \right.$$
$$\left. + t\log\left(\frac{e(m-k)}{t}\right) + \log\left(\frac{2m^2}{\delta}\right) \right].$$

## 5  Discussion

Matching pursuit is a meta-scheme for creating learning algorithms for a variety of tasks. We have presented novel techniques that make it possible to analyse this style of algorithm using a combination of compression scheme ideas and more traditional learning theory. We have shown how sparse KPCA is in fact a compression scheme and demonstrated bounds that are able to accurately guide dimension selection in some cases. We have also used the techniques to bound the performance of the kernel matching pursuit (KMP) algorithm and to reinforce the generality of the approach indicated and how the approach can be extended to a matching pursuit version of KCCA.

The results in this paper imply that the performance of any learning algorithm from the matching pursuit family can be analysed using a combination of sparse and traditional learning bounds. The bounds give a general theoretical justification of the framework and suggest potential applications of matching pursuit methods to other learning tasks such as novelty detection, ranking and so on.

## Acknowledgements

The work was sponsored by the PASCAL network of excellence and the SMART project.

## Footnotes

[1]The algorithm was proposed as a low rank kernel approximation – however the algorithm turns out to be a sparse kernel PCA (to be shown).

[2] In their book, Smola and Schölkopf redefine their kernel approximation in the same way as we have done [5], however they do not make the connection that it is a compression scheme (see Claim 1).

[3] We do not use the "59-trick" in our algorithm – although it's inclusion would be trivial and would result in the same algorithm as in [8]

## References

[1] M. Anthony. Partitioning points by parallel planes. *Discrete Mathematics*, 282:17–21, 2004.

[2] S. Floyd and M. Warmuth. Sample compression, learnability, and the Vapnik-Chervonenkis dimension. *Machine Learning*, 21(3):269–304, 1995.

[3] N. Littlestone and M. K. Warmuth. Relating data compression and learnability. Technical report, University of California Santa Cruz, Santa Cruz, CA, 1986.

[4] S. Mallat and Z. Zhang. Matching pursuit with time-frequency dictionaries. *IEEE Transactions on Signal Processing*, 41(12):3397–3415, 1993.

[5] B. Schölkopf and A. Smola. *Learning with Kernels*. MIT Press, Cambridge, MA, 2002.

[6] J. Shawe-Taylor and N. Cristianini. *Kernel Methods for Pattern Analysis*. Cambridge University Press, Cambridge, U.K., 2004.

[7] J. Shawe-Taylor, C. K. I. Williams, N. Cristianini, and J. Kandola. On the eigenspectrum of the Gram matrix and the generalization error of kernel-PCA. *IEEE Transactions on Information Theory*, 51(7):2510–2522, 2005.

[8] A. J. Smola and B. Schölkopf. Sparse greedy matrix approximation for machine learning. In *Proceedings of 17th International Conference on Machine Learning*, pages 911–918. Morgan Kaufmann, San Francisco, CA, 2000.

[9] V. N. Vapnik and A. Y. Chervonenkis. On the uniform convergence of relative frequencies of events to their probabilities. *Theory of Probability and its Applications*, 16(2):264–280, 1971.

[10] P. Vincent and Y. Bengio. Kernel matching pursuit. *Machine Learning*, 48:165–187, 2002.

[11] C. K. I. Williams and M. Seeger. Using the Nyström method to speed up kernel machines. In *Advances in Neural Information Processing Systems*, volume 13, pages 682–688. MIT Press, 2001.

